# LEARNING BY STATE RECURRENCE DETECTION

Bruce E. Rosen, James M. Goodwin[†], and Jacques J. Vidal
University of California, Los Angeles, Ca. 90024

## ABSTRACT

This research investigates a new technique for unsupervised learning of nonlinear control problems. The approach is applied both to Michie and Chambers BOXES algorithm and to Barto, Sutton and Anderson's extension, the ASE/ACE system, and has significantly improved the convergence rate of stochastically based learning automata.

Recurrence learning is a new nonlinear reward-penalty algorithm. It exploits information found during learning trials to reinforce decisions resulting in the recurrence of nonfailing states. Recurrence learning applies positive reinforcement during the exploration of the search space, whereas in the BOXES or ASE algorithms, only negative weight reinforcement is applied, and then only on failure. Simulation results show that the added information from recurrence learning increases the learning rate.

Our empirical results show that recurrence learning is faster than both basic failure driven learning and failure prediction methods. Although recurrence learning has only been tested in failure driven experiments, there are goal directed learning applications where detection of recurring oscillations may provide useful information that reduces the learning time by applying negative, instead of positive reinforcement.

Detection of cycles provides a heuristic to improve the balance between evidence gathering and goal directed search.

## INTRODUCTION

This research investigates a new technique for unsupervised learning of nonlinear control problems with delayed feedback. Our approach is compared to both Michie and Chambers BOXES algorithm[1], to the extension by Barto, et al., the ASE (Adaptive Search Element) and to their ASE/ACE (Adaptive Critic Element) system[2], and shows an improved learning time for stochastically based learning automata in failure driven tasks.

We consider adaptively controlling the behavior of a system which passes through a sequence of states due to its internal dynamics (which are not assumed to be known a priori) and due to choices of actions made in visited states. Such an adaptive controller is often referred to as a learning automaton. The decisions can be deterministic or can be made according to a stochastic rule. A learning automaton has to discover which action is best in each circumstance by producing actions and observing the resulting information.

This paper was motivated by the previous work of Barto, et al. to investigate neuronlike adaptive elements that affect and learn from their environment. We were inspired by their current work and the recent attention to neural networks and connectionist systems, and have chosen to use the cart-pole control problem[2], to enable a comparison of our results with theirs.

---

[†]Permanent address: California State University, Stanislaus; Turlock, California.

## THE CART-POLE PROBLEM

In their work on the cart-pole problem, Barto, Sutton and Anderson considered a learning system composed of an automaton interacting with an environment. The problem requires the automaton to balance a pole acting as an inverted pendulum hinged on a moveable cart. The cart travels left or right along a bounded one dimensional track; the pole may swing to the left or right about a pivot attached to the cart. The automaton must learn to keep the pole balanced on the cart, and to keep the cart within the bounds of the track. The parameters of the cart/pole system are the cart position and velocity, and the pole angle and angular velocity. The only actions available to the automaton are the applications of a fixed impulsive force to the cart in either right or left direction; one of these actions *must* be taken.

This balancing is an extremely difficult problem if there is no a priori knowledge of the system dynamics, if these dynamics change with time, or if there is no preexisting controller that can be imitated (e.g. Widrow and Smith's[3] ADALINE controller). We assumed no a priori knowledge of the dynamics nor any preexisting controller and anticipate that the system will be able to deal with any changing dynamics.

Numerical simulations of the cart-pole solution via recurrence learning show substantial improvement over the results of Barto et al., and of Michie and Chambers, as is shown in figure 1. The algorithms used, and the results shown in figure 1, will be discussed in detail below.

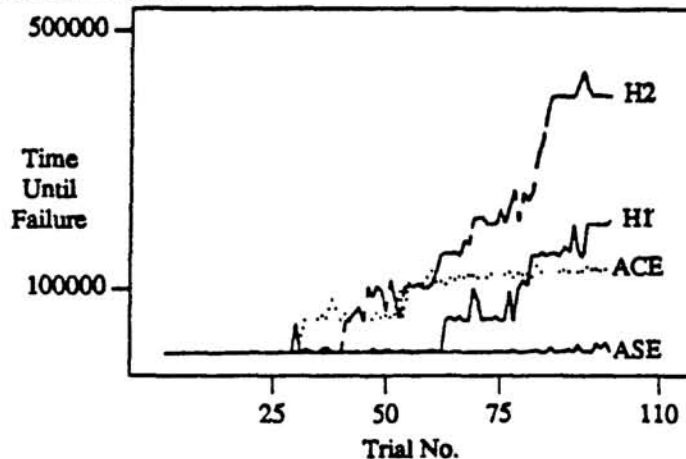

Figure 1: Performance of the ASE, ASE/ACE, Constant Recurrence (H1) and Short Recurrence (H2) Algorithms.

## THE GENERAL PROBLEM: ASSIGNMENT OF CREDIT

The cart-pole problem is one of a class of problems known as "credit assignment"[4], and in particular *temporal* credit assignment. The recurrence learning algorithm is an approach to the general temporal credit assignment problem. It is characterized by seeking to improve learning by making decisions about early actions. The goal is to find actions responsible for improved or degraded performance at a much later time.

An example is the bucket brigade algorithm[5]. This is designed to assign credit to rules in the system according to their overall usefulness in attaining their goals. This is done by adjusting the strength value (weight) of each rule. The problem is of modifying these strengths is to permit rules activated early in the sequence to result in successful actions later.

Samuels considered the credit assignment problem for his checkers playing program[6]. He noted that it is easy enough to credit the rules that combine to produce a triple jump at some point in the game; it is much harder to decide which rules active earlier were responsible for changes that made the later jump possible.

State recurrence learning assigns a strength to an individual rule or action and modifies that action's strength (while the system accumulates experience) on the basis of the action's overall usefulness in the situations in which it has been invoked. In this it follows the bucket brigade paradigm of Holland.

## PREVIOUS WORK

The problems of learning to control dynamical systems have been studied in the past by Widrow and Smith[3], Michie and Chambers[1], Barto, Sutton, and Anderson[2], and Connell[7]. Although different approaches have been taken and have achieved varying degrees of success, each investigator used the cart/pole problem as the basis for empirically measuring how well their algorithms work.

Michie and Chambers[1] built BOXES, a program that learned to balance a pole on a cart. The BOXES algorithm choose an action that had the highest average time until failure. After 600 trials (a trial is a run ending in eventual failure or by some time limit expiration), the program was able to balance the pole for 72,000 time steps. Figure 2a describes the BOXES learning algorithm. States are penalized (after a system failure) according to recency. Active states immediately preceding a system failure are punished most.

Barto, Sutton and Anderson[2] used two neuronlike adaptive elements to solve the control problem. Their ASE/ACE algorithm chose the action with the highest probability of keeping the pole balanced in the region, and was able to balance the pole for over 60,000 time steps before completion of the 100th trial.

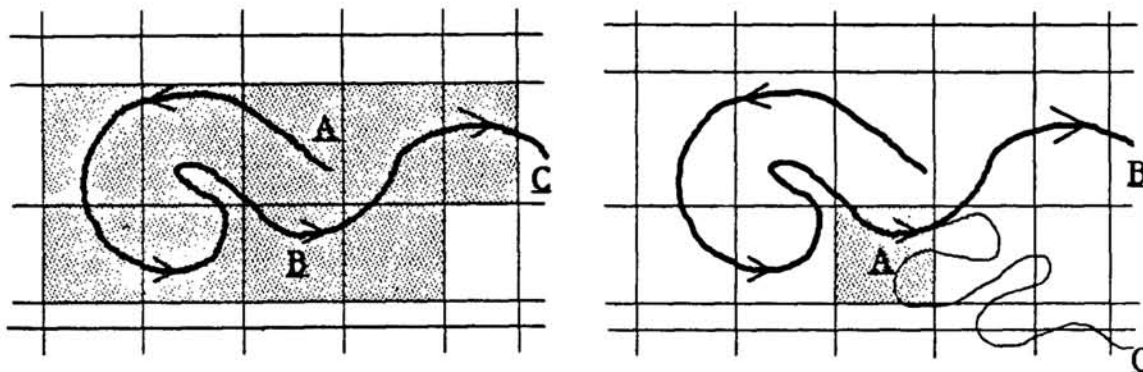

Figure 2a and 2b: The BOXES and ASE/ACE (Associative Search Elelement - Adpative Critic Element) algorithms

Figure 2a shows the BOXES (and ASE) learning algorithm paradigm When the automaton enters a failure state (C), all states that it has traversed (shaded rectangles) are punished, although state B is punished more than state A. (Failure states are those at the edges of the diagram.) Figure 2b describes the ASE/ACE learning algorithm. If a system failure occurs before a state's expected failure time, the state is penalized. If a system failure occurs after its expected failure time, the state is rewarded. State A is penalized because a failure occurred at B sooner than expected. State A's expected

failure time is the time for the automaton to traverse from state A to failure point C. When leaving state A, the weights are updated if the new state's expected failure time differs from that of state A.

Anderson[8] used a connectionist system to learn to balance the pole. Unlike the previous experiments, the system did provide well-chosen states a priori. On the average, 10,000 trials were necessary to learn to balance the pole for 7000 time steps.

Connell and Utgoff[7] developed an approach that did not depend on partitioning the state space into discrete regions. They used Shepard's function[9,10] to interpolate the degree of desirability of a cart-pole state. The system learned the control task after 16 trials. However, their system used a knowledge representation that had a priori information about the system.

## OTHER RELATED WORK

Klopf[11] proposed a more neurological class of differential learning mechanisms that correlates earlier changes of inputs with later changes of outputs. The adaptation formula used multiplies the change in outputs by the weighted sum of the absolute value of the t previous inputs weights ($\Delta w_j$), the $\tau$ previous differences in inputs ($\Delta x_j$), and the $\tau$ previous time coefficients ($c_j$).

Sutton's temporal differences (TD)[12] approach is one of a class of adaptive prediction methods. Elements of this class use the sum of previously predicted output values multiplied by the gradient and an exponentially decaying coefficient to modify the weights. Barto and Sutton [13] used temporal differences as the underlying learning procedure for classical conditioning.

## THE RECURRENCE LEARNING METHOD

### DEFINITIONS

A *state* is the set of values (or ranges) of parameters sufficient to specify the instantaneous condition of the system.

The *input decoder* groups the environmental states into equivalence classes: elements of one class have identical system responses. Every environmental input is mapped into one of n input states. (All further references to "states" assumes that the input values fall into the discrete ranges determined by the decoder, unless otherwise specified.)

States returned to after visiting one or more alternate states *recur*.

An *action* causes the modification of system parameters, which may change the system state. However, no change of state need occur, since the altered parameter values may be decoded within the same ranges.

A *weight*, w(t), is associated with each action for each state, with the probability of an allowed action dependent on the current value of its weight.

A *rule* determines which of the allowable actions is taken. The rule is not deterministic. It chooses an action stochastically, based on the weights.

Weight *changes*, $\Delta w(t)$, are made to reduce the likelihood of choosing an action which will cause an eventual failure. These changes are made based on the idea that the previous action of an element, when presented with input x(t), had some influence in causing a similar pattern to occur again. Thus, weight changes are made to increase the likelihood that an element produces the same action f(t) when patterns similar to x(t) occur in the future.

For example, consider the classic problem of balancing a pole on a moving cart. The *state* is specified by the positions and velocities of both the cart and the pole. The allowable *actions* are fixed velocity increments to the right or to the left, and the *rule* determines which action to take, based on the current weights.

## THE ALGORITHM

The recurrence learning algorithm presented here is a nonlinear reward-penalty method[14]. Empirical results show that it is successful for stationary environments. In contrast to other methods, it also may be applicable to nonstationary environments'. Our efforts have been to develop algorithms that reward decision choices that lead the controller/environment to quasi-stable cycles that avoid failure (such as limit cycles, converging oscillations and absorbing points).

Our technique exploits recurrence information obtained during learning trials. The system is rewarded upon return to a previous state, however weight changes are only permitted when a state transition occurs. If the system returns to a state, it has avoided failure. A recurring state is rewarded. A sequence of recurring states can be viewed as evidence for a (possibly unstable) cycle. The algorithm forms temporal "cause and effect" associations.

To optimize performance, dynamic search techniques must balance between choosing a search path with known solution costs, and exploring new areas of the search space to find better or cheaper solutions. This is known as the two armed bandit problem[15], i.e. given a two handed slot machine with one arm's *observed* reward probabilities higher than the other, one should not exclude playing with the arm with the lesser payoff. Like the ASE/ACE system, recurrence learning learns while searching, in contrast to the BOXES and ASE algorithms which learn only upon failure.

## RANGE DECODING

In our work, as in Barto and others, the real valued input parameters are analyzed as members of ranges. This reduces computing resource demands. Only a limited number of ranges are allowed for each parameter. It is possible for these ranges to overlap, although this aspect of range decoding is not discussed in this paper, and the ranges were considered nonoverlapping. When the parameter value falls into one of the ranges that range is *active*. The specification of a state consists of one of the active ranges for each of the parameters. If the ranges do not overlap, then the set of parameter values specify one unique state; otherwise the set of parameter values may specify several states. Thus, the parameter values at any time determine one or several active states $S_i$ from the set of n possible states.

The value of each environmental parameter falls into one of a number of ranges, which may be different for different parameters. A state is specified by the active range for each parameter.

The set of input parameter values are decoded into one (or more) of n ranges $S_i$, $0 <= i <= n$. For this problem, boolean values are used to describe the activity level of a state $S_i$. The activity value of a state is 1 if the state is active, or 0 if it is inactive.

## ACTION DECISIONS

Our model is the same as that of the BOXES and ASE/ACE systems, where only one input (and state) is active at any given time. All states were nonoverlapping and mutually exclusive, although there was no reason to preclude them from overlapping

other than for consistency with the two previous models. In the ASE/ACE system and in ours as well, the output decision rule for the controller is based on the weighted sum of its inputs plus some stochastic noise. The action (output) decision of the controller is either +1 or -1, as given by:

$$y_i(t) = f\left( \sum_{i=1}^{n} w_i(t) \, x_i(t) + noise(t) \right) \qquad (1)$$

where

$$f(z) = \begin{bmatrix} +1 \text{ if } z \geq 0 \\ -1 \text{ if } z < 0 \end{bmatrix} \qquad (2)$$

and noise is a real randomly (Gaussian) distributed value with some mean $\mu$ and variance $\sigma^2$. An output, $f(z)$, for the car/pole controller is interpreted as a push to the left if $f(z) = -1$ or to the right if $f(z) = +1$.

## RECURRENCE LEARNING

The goal of the recurrence learning algorithm is to avoid failure by moving toward states that are part of cycles if such states exist, or quasi-stable oscillations if they don't. This concept can be compared to juggling. As long as all the balls are in the air, the juggler is judged a success and rewarded. No consideration is given to whether the balls are thrown high or low, left or right; the controller, like the juggler, tries for the most stable cycles. Optimum performance is not demanded from recurrence learning.

Two heuristics have been devised. The fundamental basis of each of them is to reward a state for being repeatedly visited (or repeatedly activated). The first heuristic is to reward a state when it is revisited, as part of a cycle in which no failure had occurred. The second heuristic augments the first by giving more reward to states which participate in shorter cycles. These heuristics are discussed below in detail.

HEURISTIC H1: *If a state has been visited more than once during one trial, reward it by reinforcing its weight.*

## RATIONALE

This heuristic assumes that states that are visited more than once have been part of a cycle in which no failure had occurred. The action taken in the previous visit is assumed to have had some influence on the recurrence. By reinforcing a weight upon state revisitation, it is assumed to increase the likelihood that the cycle will occur again. No assumptions are made as to whether other states were responsible for the cycle.

## RESTRICTION

An action may not immediately cause the environment to change to a different state. There may be some delay before a transition, since small changes of parameters may be decoded into the same input ranges, and hence the same state. This inertia is incorporated into the heuristics. When the same state appears twice in succession, its weight is not reinforced, since that would assume that the *action*, rather than inertia, directly caused the state's immediate recurrence.

## THE RECURRENCE EQUATIONS

The recurrence learning equations stem in part from the weight alteration formula used in the ASE system. The weight of a state is a sum of its previous weight, and the product of the learning rate ($\alpha$), the reward (r), and the state's eligibility (e).

$$w_i(t+1) = w_i(t) + \alpha r(t)e_i(t) \qquad r(t) \in \{-1,0\} \qquad (3)$$

The eligibility index $e_i(t)$ is an exponentially decaying trace function.

$$e_i(t+1) = \beta e_i(t) + (1- \beta)y_i(t)x_i(t) \qquad (4)$$

where $0<=\beta<=1$, $x_i \in \{0,1\}$, and $y_i \in \{-1,1\}$. The output value $y_i$ is the last output decision, and $\beta$ determines the decay rate.
The reward function is:

$$r(t) = \left\{ \begin{array}{ll} -1 & \text{when the system fails at time t} \\ 0 & \text{otherwise} \end{array} \right\} \qquad (5)$$

## REINFORCEMENT OF CYCLES

Equations (1) through (5) describe the basic ASE system. Our algorithm extends the weight updating procedure as follows:

$$w_i(t+1) = w_i(t) + \alpha r(t)e_i(t) + \alpha_2 r_2(t)e_{2,i}(t) \qquad (6)$$

The term $\alpha r(t)e_i(t)$ is the same as in (3), providing failure reinforcement. The term $\alpha_2 r_2(t)e_{2,i}(t)$ provides reinforcement for success. When state i is eligible (by virtue of $x_i > 0$), there is a weight change by the amount: $\alpha_2$ multiplied by the reward value, $r_2(t)$, and the current eligibility $e_{2,i}(t)$. For simplicity, the reward value, $r_2(t)$, may be taken to be some positive constant, although it need not be; any environmental feedback, yielding a reinforcement value as a function of time could be used instead. The second eligibility function $e_{2,i}(t)$ yields one of three constant values for H1: $-\beta_2$, 0, or $\beta_2$ according to formula (7) below:

$$e_{2,i}(t) = \left\{ \begin{array}{ll} 0 & \text{if } t-t_{i,last} = 1 \text{ or } t_{i,last} = 0 \\ \beta_2 x_i(t)y(t_{i,last}) & \text{otherwise} \end{array} \right\} \qquad (7)$$

where $t_{i,last}$ is the last time that state was active. If a state has not previously been active (i.e. $x_i(t) = 0$ for all t) then $t_{i,last}=0$. As the formula shows, $e_{2,i}(t) = 0$ if the state has not been previously visited or if no state transition occurred in the last time step; otherwise, $e_{2,i}(t) = \beta_2 x_i(t)y(t_{i,last})$.
The direction (increase or decrease) of the weight change due to the final term in (6) is that of the last action taken, $y(t_{i,last})$.

Heuristic **H1** is called *constant recurrence learning* because the eligibility function is designed to reinforce *any* cycle.

**HEURISTIC H2:** *Reward a short cycle more than a longer one.*

Heuristic **H2** is called *short recurrence learning* because the eligibility function is designed to reinforce *shorter* cycle more than longer cycles.

## REINFORCEMENT OF SHORTER CYCLES

The basis of the second heuristic is the conjecture that short cycles converge more easily to absorbing points than long ones, and that long cycles diverge more easily than shorter ones, although any cycle can "grow" or diverge to a larger cycle. The following extension to the our basic heuristic is proposed.
The formula for the recurrence eligibility function is:

$$e_{2,i}(t) = \begin{cases} 0 & \text{if } t\text{-}t_{i,last} = 1 \text{ or } t_{i,last} = 0 \\ \dfrac{\beta_2}{(\beta_2+t\text{-}t_{i,last})} x_i(t)\, y(t_{i,last}) & \text{otherwise} \end{cases} \qquad (8)$$

The current eligibility function $e_{2,i}(t)$ is similar to the previous failure eligibility function in (7); however, $e_{2,i}(t)$ reinforces shorter cycles more, because the eligibility decays with time. The value returned from $e_{2,i}(t)$ is inversely proportional to the period of the cycle from $t_{i,last}$ to t. **H2** reinforces converging oscillations; the term $\alpha_2 r_2(t) e_{2,i}(t)$ in (6) ensures weight reinforcement for returning to an already visited state.

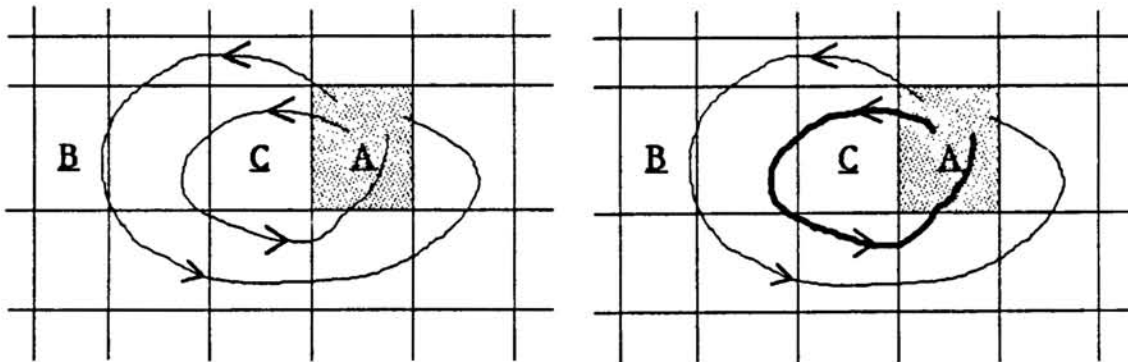

Figure 3a and 3b: The Constant Recurrence algorithm and Short Recurrence algorithms

Figure 3A shows the Constant Recurrence algorithm (H1). A state is rewarded when it is reactivated by a transition from another state. In the example below, state A is reward by a constant regardless of weather the cycle traversed states B or C. Figure 3b describes the Short Recurrence algorithm (H2). A state is rewarded according to the difference between the current time and its last activation time. Small differences are rewarded more than large differences In the example below, state A is rewarded more

when the cycle (through state C) traverses the states shown by the dark heavy line rather than when the cycle (through state B) traverses the lighter line, since state A recurs sooner when traversing the darker line.

## SIMULATION RESULTS

We simulated four algorithms: ASE, ASE/ACE and the two recurrence algorithms. Each experiment consisted of ten runs of the cart-pole balancing task, each consisting of 100 trials. Each trial lasted for 500,000 time steps or until the cart-pole system failed (i.e. the pole fell or the cart went beyond the track boundaries). In an effort to conserve cpu time, simulations were also terminated when the system achieved two consecutive trials each lasting for over 500,000 time steps; all remaining trials were assumed to also last 500,000 time steps. This assumption was reasonable: the resulting weight space causes the controller to become deterministic regardless of the influence of stochastic noise. Because of the long time require to run simulations, no attempts were made to optimize parameters of the algorithm.

As in Barto[2], each trial began with the cart centered, and the pole upright. No assumptions were made as to the state space configuration, the desirability of the initial states, or the continuity of the state space.

The first experiment consisted of failure and recurrence reward learning. The ASE failure learning runs averaged 1578 time steps until failure after 100 trials*. Next, the predictive ASE/ACE system was run as a comparative metric, and it was found that this method caused the controller to average 131,297 time steps until failure; the results are comparable to that described by Barto, Sutton and Anderson.

In the next experiment, short recurrence learning system was added to the ASE system. Again, ten 100 trial learning session were executed. On the average, the short recurrence learning algorithm ran for over 400,736 time steps after 100th trial, bettering the ASE/ACE system by 205%.

In the final experiment, constant recurrence learning with the ASE system was simulated. The constant recurrence learning eliminated failure after only 207,562 time steps.

Figure 1 shows the ASE, ASE/ACE, Constant recurrence learning (H1) and Short recurrence learning (H2) failure rates averaged over 10 simulation runs.

## DISCUSSION

Detection of cycles provides a heuristic for the "two armed bandit" problem to decide between evidence gathering, and goal directed search. The algorithm allows the automaton to search outward from the cycle states (states with high probability of revisitation) to the more unexplored search space. The rate of exploration is proportional to the recurrence learning parameter $\alpha_2$; as $\alpha_2$ is decreased, the influence of the cycles governing the decision process also decreases and the algorithm explores more of the search space that is not part of any cycle or oscillation path.

## THE FUTURE

Our future experiments will study the effects of rewarding predictions of cycle lengths in a manner similar to the prediction of failure used by the ASE/ACE system. The effort will be to minimize the differences of predicted time of cycles in order to predict their period. Results of this experiment will be shown in future reports. We hope to show that this recurrence prediction system is generally superior to either the ASE/ACE predictive system or the short recurrence system operating alone.

## CONCLUSION

This paper presented an extension to the failure driven learning algorithm based on reinforcing decisions that cause an automaton to enter environmental states more than once. The controller learns to synthesize the best values by reinforcing areas of the search space that produce recurring state visitation. Cycle states, which under normal failure driven learning algorithms do not learn, achieve weight alteration from success. Simulations show that recurrence reward algorithms show improved overall learning of the cart-pole task with a substantial decrease in learning time.

## Footnotes

* However, there was a relatively large degree of variance in the final trials. The last 10 trails (averaged over each of the 10 simulations) ranged from 607 to 15,459 time steps until failure

## REFERENCES

1.  D. Michie and R. Chambers, *Machine Intelligence*, E. Dale and D. Michie, Ed.: (Oliver and Boyd, Edinburgh, 1968), p. 137.
2.  A. Barto, R. Sutton, and C. Anderson, *Coins Tech. Rept.*, No. 82-20, 1982.
3.  B. Widrow and F. Smith, in *Computer and Information Sciences*, J. Tou and R. Wilcox Eds., (Clever Hume Press, 1964).
4.  M. Minsky, in *Proc. IRE*, **49**, 8, (1961).
5.  J. Holland, in *Proc. Int. Conf., Genetic Algs. and their Appl.*, 1985, p. 1.
6.  A. Samuel, *IBM Journ. Res.and Dev.* **3**, 211, (1959)
7.  M. Connell and P. Utgoff, in *Proc. AAAI-87* (Seattle, 1987), p. 456.
8.  C. Anderson, *Coins Tech. Rept.*, No. 86-50: Amherst, MA. 1986.
9.  R. Barnhill, in *Mathematical Software III*, (Academic Press, 1977).
10. L. Schumaker, in *Approximation Theory II.* (Academic Press, 1976).
11. A. H. Klopf, in *IEEE Int. Conf. Neural Networks,*, June 1987.
12. R. Sutton, *GTE Tech. Rept.TR87-509.1*, GTE Labs. Inc., Jan. 1987
13. R. Sutton and A. G. Barto, *Tech. Rept. TR87-5902.2* March 1987
14. A. Barto and P. Anandan, *IEEE Trans. SMC* **15**, 360 (1985).
15. M. Sato, K. Abe, and H. Takeda, *IEEE Trans.SMC* **14** , 528 (1984).
